# Self-similarity properties of natural images

ANTONIO TURIEL,* GERMÁN MATO! NÉSTOR PARGA ‡
*Departamento de Física Teórica . Universidad Autónoma de Madrid*
*Cantoblanco, 28049 Madrid, Spain*
and JEAN-PIERRE NADAL§
*Laboratoire de Physique Statistique de l'E.N.S. ¶ Ecole Normale Supérieure*
*24, rue Lhomond, F-75231 Paris Cedex 05, France*

## Abstract

Scale invariance is a fundamental property of ensembles of natural images [1]. Their non Gaussian properties [15, 16] are less well understood, but they indicate the existence of a rich statistical structure. In this work we present a detailed study of the marginal statistics of a variable related to the edges in the images. A numerical analysis shows that it exhibits extended self-similarity [3, 4, 5]. This is a scaling property stronger than self-similarity: all its moments can be expressed as a power of any given moment. More interesting, all the exponents can be predicted in terms of a multiplicative log-Poisson process. This is the very same model that was used very recently to predict the correct exponents of the structure functions of turbulent flows [6]. These results allow us to study the underlying multifractal singularities. In particular we find that the most singular structures are one-dimensional: the most singular manifold consists of sharp edges.

Category: *Visual Processing.*

## 1  Introduction

An important motivation for studying the statistics of natural images is its relevance for the modeling of the visual system. In particular, the epigenetic development

¶Laboratoire associé au C.N.R.S. (U.R.A. 1306), à l'ENS, et aux Universités Paris VI et Paris VII.

could lead to the adaptation of visual processing to the statistical regularities in the visual scenes [8, 9, 10, 11, 12, 13]. Most of these predictions on the development of receptive fields have been obtained using a gaussian description of the environment contrast statistics. However non Gaussian properties like the ones found by [15, 16] could be important. To gain further insight into non Gaussian aspects of natural scenes we investigate the self similarity properties of an edge type variable [14].

Scale invariance in natural images is a well-established property. In particular it appears as a power law behaviour of the power spectrum of luminosity contrast: $S(f) \propto \frac{1}{|f|^{2-\eta}}$ (the parameter $\eta$ depends on the particular images that has been included in the dataset). A more detailed analysis of the scaling properties of the luminosity contrast was done by [15, 16]. These authors noted the possible analogy between the statistics of natural images and turbulent flows. There is however no model to explain the scaling behaviour that they observed.

On the other hand, a large amount of effort has been put to understand the statistics of turbulent flows and to develop predictable models (see e.g. [17]). Qualitative and quantitative theories of fully developed turbulence elaborate on the original argument of Kolmogorov [2]. The cascade of energy from one scale to another is described in terms of local energy dissipation per unit mass within a box of linear size $r$. This quantity, $\epsilon_r$, is given by:

$$\epsilon_r(\mathbf{x}) \propto \int_{|\mathbf{x}-\mathbf{x}'|<r} d\mathbf{x}' \sum_{ij} [\partial_i v_j(\mathbf{x}') + \partial_j v_i(\mathbf{x}')]^2 \qquad (1)$$

where $v_i(\mathbf{x})$ is the $i$th component of the velocity at point $\mathbf{x}$. This variable has *Self-Similarity* (SS) properties that is, there is a range of scales $r$ (called the inertial range) where:

$$<\epsilon_r^p> \propto r^{\tau_p}, \qquad (2)$$

here $<\epsilon_r^p>$ denotes the $p$th moment of the energy dissipation marginal distribution. A more general scaling relation, called *Extended Self-Similarity* (ESS) has been found to be valid in a much larger scale domain. This relation reads

$$<\epsilon_r^p> \propto <\epsilon_r^q>^{\rho(p,q)} \qquad (3)$$

where $\rho(p, q)$ is the ESS exponent of the $p$th moment with respect to the $q$th moment. Let us notice that if SS holds then $\tau_p = \tau_q \rho(p, q)$. In the following we will refer all the moments to $<\epsilon_r^2>$.

## 2 The Local Edge Variance

For images the basic field is the contrast $c(\mathbf{x})$, that we define as the difference between the luminosity and its average. By analogy with the definition in eq. (1) we will consider a variable that accumulates the value of the variation of the contrast. We choose to study two variables, defined at position $\mathbf{x}$ and at scale $r$. The variable $\epsilon_{h,r}(\mathbf{x})$ takes contributions from edges transverse to a *horizontal* segment of size $r$:

$$\epsilon_{h,r}(\mathbf{x}) = \frac{1}{r} \int_{x_1}^{x_1+r} \left( \frac{\partial c(\mathbf{x}')}{\partial y} \right)^2 \Bigg|_{\mathbf{x}'=\{y,x_2\}} dy \qquad (4)$$

A vertical variable $\epsilon_{v,r}(\mathbf{x})$ is defined similarly integrating along the vertical direction.

We will refer to the value of the derivative of the contrast along a given direction as an edge transverse to that direction. This is justified in the sense that in the presence of borders this derivative will take a great value, and it will almost vanish

if evaluated inside an almost-uniformly illuminated surface. Sharp edges will be the maxima of this derivative. According to its definition, $\epsilon_{l,r}(\mathbf{x})$ ( $l = h, v$ ) is the *local linear edge variance* along the direction $l$ at scale $r$. Let us remark that edges are well known to be important in characterizing images. A recent numerical analysis suggests that natural images are composed of statistically independent edges [18].

We have analyzed the scaling properties of the local linear edge variances in a set of 45 images taken into a forest, of $256 \times 256$ pixels each (the images have been provided to us by D. Ruderman; see [16] for technical details concerning them). An analysis of the image resolution and of finite size effects indicates the existence of upper and lower cut-offs. These are approximately $r = 64$ and $r = 8$, respectively. First we show that SS holds in a range of scales $r$ with exponents $\tau_{h,p}$ and $\tau_{v,p}$. This is illustrated in Fig. (1) where the logarithm of two moments of horizontal and vertical local edge variances are plotted as a function of $\ln r$; we see that SS holds, but not in the whole range.
ESS holds in the whole considered range; two representative graphs are shown in Fig. (2). The linear dependence of $\ln < \epsilon_{l,r}^p >$ vs $\ln < \epsilon_{l,r}^2 >$ is observed in both the horizontal ($l = h$) and the vertical ($l = v$) directions. This is similar to what is found in turbulence, where this property has been used to obtain a more accurate estimation of the exponents of the structure functions (see e.g. [17] and references therein). The exponents $\rho_h(p,2)$ and $\rho_v(p,2)$, estimated with a least squares regression, are shown in Fig. (3) as a function of $p$. The error bars refer to the statistical dispersion. From figs. (1-3) one sees that the horizontal and vertical directions have similar statistical properties. The SS exponents differ, as can be seen in Fig(1); but, surprisingly, ESS not only holds in both directions, but it does it with the *same* ESS exponents, i.e. $\rho_h(p,2) \sim \rho_v(p,2)$.

## 3   ESS and multiplicative processes

Let us now consider scaling models to predict the $p$-dependence of the ESS exponents $\rho_l(p,2)$. (Since ESS holds, the SS exponents $\tau_{l,p}$ can be obtained from the $\rho_l(p,2)'s$ by measuring $\tau_{l,2}$). The simplest scaling hypothesis is that, for a random variable $\epsilon_r(\mathbf{x})$ observed at the scale $r$ (such as $\epsilon_{l,r}(\mathbf{x})$), its probability distribution $\bar{P}_r(\epsilon_r(\mathbf{x}) = \epsilon)$ can be obtained from any other scale $L$ by

$$\bar{P}_r(\epsilon) = \frac{1}{\alpha(r,L)} \bar{P}_L \left( \frac{\epsilon}{\alpha(r,L)} \right) \tag{5}$$

From this one derives easily that $\alpha(r,L) = [\frac{<\epsilon_r^p>}{<\epsilon_L^p>}]^{1/p}$ (for any $p$) and $\rho(p,2) \propto p$; if SS holds, $\tau_p \propto p$: for turbulent flows this corresponds to the Kolmogorov prediction for the SS exponents [2]. Fig (3) shows that this naive scaling is violated.
This discrepancy becomes more dramatic if eq. (5) is expressed in terms of a normalized variable. Taking $\epsilon_r^\infty = \lim_{p\to\infty} < \epsilon_r^{p+1} > / < \epsilon_r^p >$ ( that can be shown to be the maximum value of $\epsilon_r$, which in fact is **finite** ) the new variable is defined as $f_r = \epsilon_r/\epsilon_r^\infty$; $0 < f_r < 1$. If $P_r(f)$ is the distribution of $f_r$, the scaling relation eq.(5) reads $P_r(f) = P_L(f)$; this identity does not hold as can be seen in Fig. (4). A way to generalize this scaling hypothesis is to say that $\alpha$ is no longer a constant as in eq. (5), but an stochastic variable. Thus, one has for $P_r(f)$ :

$$P_r(f) = \int G_{rL}(\ln \alpha) \frac{1}{\alpha} P_L \left( \frac{f}{\alpha} \right) d\ln \alpha \tag{6}$$

This scaling relation has been first introduced in the context of turbulent flows [6, 19, 7]. Eq. (6) is an integral representation of ESS with general (not necessarily

linear) exponents: once the kernel $G_{rL}$ is chosen, the $\rho(p,2)$'s can be predicted. It can also be phrased in terms of multiplicative processes [20, 21] : now $f_r = \alpha f_L$, where the factor $\alpha$ itself becomes a stochastic variable determined by the kernel $G_{rL}(\ln \alpha)$. Since the scale $L$ is arbitrary (scale $r$ can be reached from any other scale $L'$) the kernel must obey a composition law, $G_{rL'} \otimes G_{L'L} = G_{rL}$. Consequently $f_r$ can be obtained through a cascade of infinitesimal processes $G_\delta \equiv G_{r,r+\delta r}$. Specific choices of $G_\delta$ define different models of ESS. The She-Leveque (SL) [6] model corresponds to a simple process such that $\alpha$ is 1 with probability $1-s$ and is a constant $\beta$ with probability $s$. One can see that $s = \frac{1}{1-\beta^2} \ln(\frac{<f_{r+\delta r}^2>}{<f_r^2>})$ and that this stochastic process yields a log-Poisson distribution for $\alpha$ [22]. It also gives ESS with exponents $\rho(p,q)$ that is expressed in terms of the parameter $\beta$ as follows [6]:

$$\rho(p,q) = \frac{1 - \beta^p - (1-\beta)p}{1 - \beta^q - (1-\beta)q} \tag{7}$$

We can now test this models with the ESS exponents obtained with the image data set. The resulting fit for the SL model is shown in Fig. (3). Both the vertical and horizontal ESS exponents can be fitted with $\beta = 0.50 \pm 0.03$.

The integral representation of ESS can also be directly tested on the probability distributions evaluated from the data. In Fig. (4) we show the prediction for $P_r(f)$ obtained from $P_L(f)$ using eq. (6), compared with the actual $P_r(f)$.

The parameter $\beta$ allows us to predict all the ESS exponents $\rho(p,2)$. To obtain the SS exponents $\tau_p$ we need another parameter. This can be chosen e.g. as $\tau_2$ or as the asymptotic exponent $\Delta$, given by $\epsilon_r^\infty \propto r^{-\Delta}$, $r \gg 1$; we prefer $\Delta$. As $\tau_p = \tau_2 \rho(p,2)$, then from the definition of $\epsilon_r^\infty$ one can see that $\Delta = -\frac{\tau_2}{1-\beta}$. A least square fit of $\tau_p$ was used to determine $\Delta$, obtaining $\Delta_h = 0.4 \pm 0.2$ for the horizontal variable and $\Delta_v = 0.5 \pm 0.2$. for the vertical one.

## 4   Multifractal analysis

Let us now partition the image in sets of pixels with the same singularity exponent $h$ of the local edge variance: $\epsilon_r \propto r^h$. This defines a multifractal with dimensions $D(h)$ given by the Legendre transform of $\tau_p$ (see e.g. [17]): $D(h) = inf_p\{ph+d-\tau_p\}$, where $d = 2$ is the dimension of the images. We are interested in the most singular of these manifolds; let us call $D_\infty$ its dimension and $h_{min}$ its singularity exponent. Since $\epsilon_r^\infty$ is the maximum value of the variable $\epsilon_r$, the most singular manifold is given by the set of points where $\epsilon_r = \epsilon_r^\infty$, so $h_{min} = -\Delta$. Using again that $\tau_p = -\Delta(1-\beta)\rho(p,2)$ with $\rho(p,2)$ given by the SL model, one has $D_\infty = d - \frac{\Delta}{(1-\beta)}$. From our data we obtain $D_{\infty,h} = 1.3 \pm 0.3$ and $D_{\infty,v} = 1.1 \pm 0.3$. As a result we can say that $D_{\infty,h} \sim D_{\infty,v} \sim 1$: the most singular structures are almost one-dimensional. This reflects the fact that the most singular manifold consists of sharp edges.

## 5   Conclusions

We insist on the main result of this work, which is the existence of non trivial scaling properties for the local edge variances. This property appears very similar to the one observed in turbulence for the local energy dissipation. In fact, we have seen that the SL model predicts all the relevant exponents and that, in particular, it describes the scaling behaviour of the sharpest edges in the image ensemble. It would also be interesting to have a simple generative model of images which - apart

from having the correct power spectrum as in [23] - would reproduce the self-similar properties found in this work.

## Acknowledgements
We are grateful to Dan Ruderman for giving us his image data base. We warmly thank Bernard Castaing for very stimulating discussions and Zhen-Su She for a discussion on the link between the scaling exponents and the dimension of the most singular structure. We thank Roland Baddeley and Patrick Tabeling for fruitful discussions. We also acknowledge Nicolas Brunel for his collaboration during the early stages of this work. This work has been partly supported by the French-Spanish program "Picasso" and an E.U. grant CHRX-CT92-0063.

## Footnotes

*e-mail: amturiel@delta.ft.uam.es

†e-mail: matog@cab.cnea.edu.ar

‡To whom correspondence should be addressed. e-mail: parga@delta.ft.uam.es

§e-mail: nadal@lps.ens.fr

## References

[1] Field D. J., *J. Opt. Soc. Am.* **4** 2379-2394 (1987).

[2] Kolmogorov, *Dokl. Akad. Nauk. SSSR* **30**, 301-305 (1941).

[3] Benzi R., Ciliberto S., Baudet C., Ruiz Chavarria G. and Tripiccione C., *Europhys. Lett.* **24** 275-279 (1993)

[4] Benzi, Ciliberto, Tripiccione, Baudet, Massaioli, and Succi, *Phys. Rev. E* **48**, R29 (1993)

[5] Benzi, Ciliberto, Baudet and Chavarria *Physica D* **80** 385-398 (1995)

[6] She and Leveque, *Phys. Rev. Lett.* **72**, 336-339 (1994).

[7] Castaing, *J. Physique II, France* **6**, 105-114 (1996)

[8] Barlow H. B., in *Sensory Communication* (ed. Rosenblith W.) pp. 217. (M.I.T. Press, Cambridge MA, 1961).

[9] Laughlin S. B., *Z. Naturf.* **36** 910-912 (1981).

[10] van Hateren J.H. *J. Comp. Physiology A* **171** 157-170, 1992.

[11] Atick J. J. *Network* **3** 213-251, 1992.

[12] Olshausen B.A. and Field D. J., *Nature* **381**, 607-609 (1996).

[13] Baddeley R., *Cognitive Science*, in press (1997).

[14] Turiel A., Mato G., Parga N. and Nadal J.-P., to appear in *Phys. Rev. Lett.*, 1998.

[15] Ruderman D. and Bialek, *Phys. Rev. Lett.* **73**, 814 (1994)

[16] Ruderman D., *Network* **5**, 517-548 (1994)

[17] Frisch U., Turbulence, Cambridge Univ. Press (1995).

[18] Bell and Sejnowski, *Vision Research* **37** 3327-3338 (1997).

[19] Dubrulle B., *Phys. Rev. Lett.* **73** 959-962 (1994)

[20] Novikov, *Phys. Rev. E* **50**, R3303 (1994)

[21] Benzi, Biferale, Crisanti, Paladin, Vergassola and Vulpiani, *Physica D* **65**, 352-358 (1993).

[22] She and Waymire, *Phys. Rev. Lett.* **74**, 262-265 (1995).

[23] Ruderman D., *Vision Research* **37** 3385-3398 (1997).

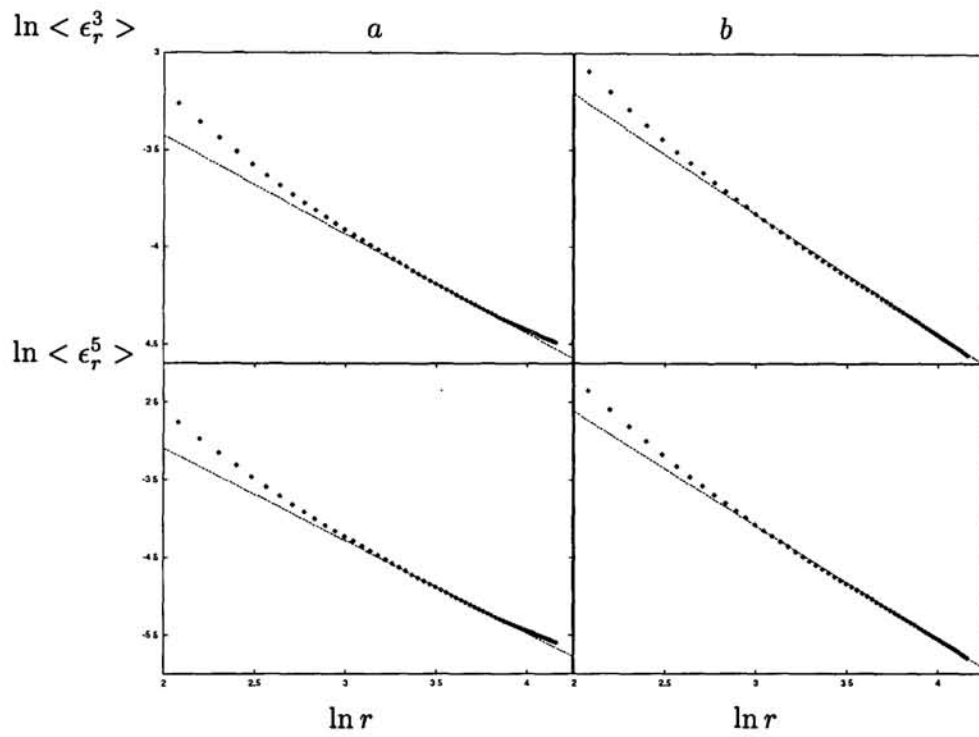

Figure 1: Test of SS. We plot $\ln < \epsilon_{l,r}^p >$ vs. $\ln r$ for $p = 3$ and 5; $r$ from 8 to 64 pixels. a) horizontal direction, $l = h$. b) vertical direction, $l = v$.

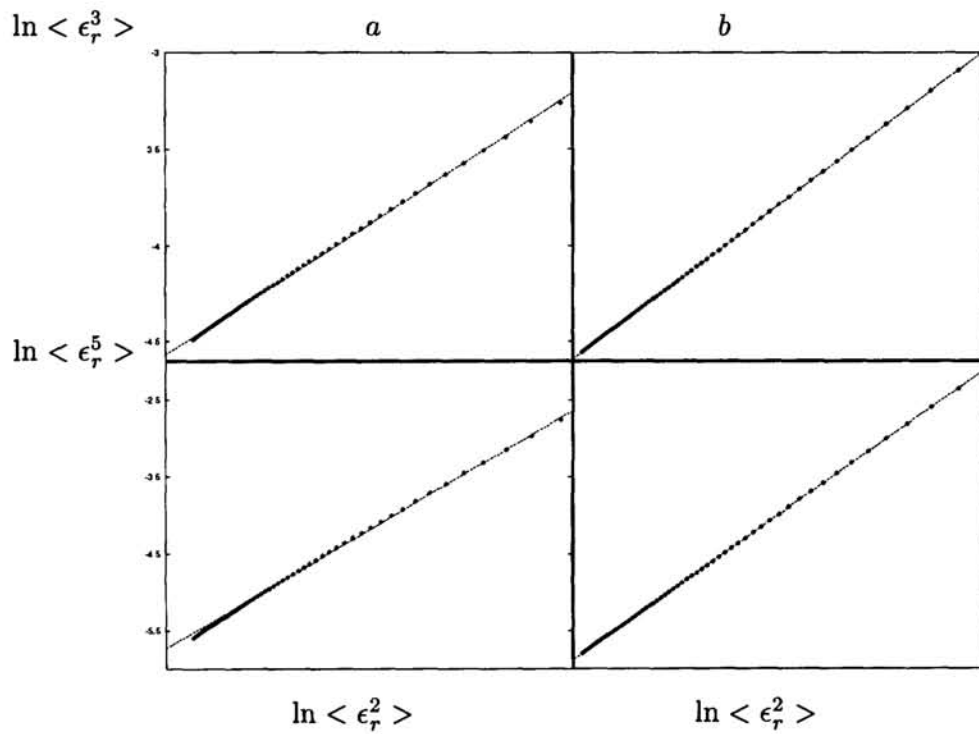

Figure 2: Test of ESS. We plot $\ln < \epsilon_{l,r}^p >$ vs. $\ln < \epsilon_{l,r}^2 >$ for p=3, 5; $r$ from 8 to $r = 64$ pixels. a) horizontal direction, $l = h$. b) vertical direction, $l = v$.

$\rho(p,2)$

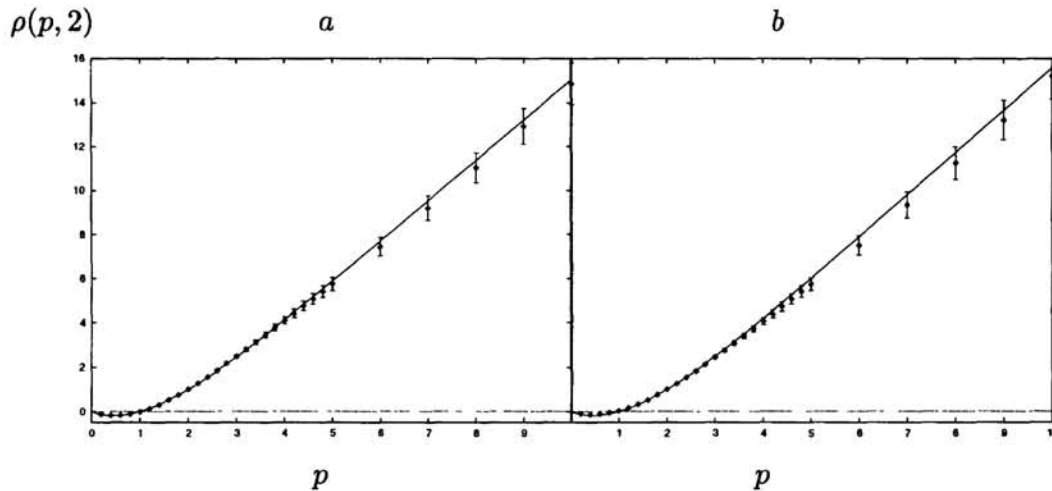

Figure 3: ESS exponents $\rho(p,2)$, for the vertical and horizontal variables. a) horizontal direction, $\rho_h(p,2)$. b) vertical direction, $\rho_v(p,2)$. The solid line represents the fit with the SL model. The best fit is obtained with $\beta_v \sim \beta_h \sim 0.50$.

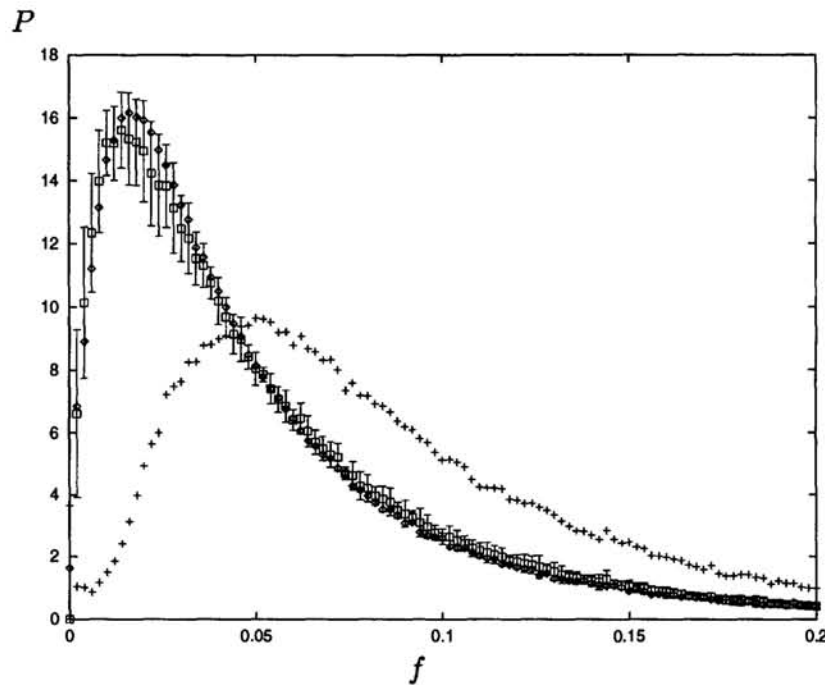

Figure 4: Verification of the validity of the integral representation of ESS, eq.(6) with a log-Poisson kernel, for horizontal local edge variance. The largest scale is $L = 64$. Starting from the histogram $P_L(f)$ (denoted with crosses), and using a log-Poisson distribution with parameter $\beta = 0.50$ for the kernel $G_{rL}$, eq.(6) gives a prediction for the distribution at the scale $r = 16$ (squares). This has to be compared with the direct evaluation of $P_r(f)$ (diamonds). Similar results hold for other pairs of scales. Although not shown in the figure, the test for vertical case is as good as for horizontal variable.